# HIGHER ORDER RECURRENT NETWORKS & GRAMMATICAL INFERENCE

C. L. Giles*,  G. Z. Sun,  H. H. Chen,  Y. C. Lee,  D. Chen
Department of Physics and Astronomy
and
Institute for Advanced Computer Studies
University of Maryland, College Park, MD 20742
* NEC Research Institute
4 Independence Way, Princeton, N.J. 08540

## ABSTRACT

A higher order single layer recursive network  easily learns to simulate  a deterministic finite  state machine and recognize regular grammars.  When an enhanced version of this neural net state machine is connected through a common error term to an external analog stack memory, the combination can be interpreted as a neural net pushdown automata.  The neural net finite state machine  is given the primitives, push and pop, and is able to read the top of the stack.  Through a gradient descent learning rule  derived  from  the  common  error function, the hybrid network learns to effectively use the stack actions to manipulate the stack memory and  to learn simple context-free grammars.

## INTRODUCTION

Biological networks readily and easily process temporal information; artificial neural networks should do the same.  Recurrent neural network models permit the encoding and learning of temporal sequences.  There are many recurrent neural net models, for example see  [Jordan 1986, Pineda 1987, Williams & Zipser 1988].  Nearly all encode the current state representation of the models in the activity of the neuron and the next state is determined by the current state and input. From an automata perspective, this dynamical structure is a state machine. One formal model of sequences and machines that generate and recognize them are formal grammars and their respective automata. These models formalize some of the foundations of computer science.  In the Chomsky hierarchy of formal grammars [Hopcroft & Ullman 1979] the simplest level of complexity is defined by the finite state machine and its regular grammars. {All machines

and grammars described here are deterministic.)   The next level of complexity is described by pushdown automata and their associated context-free grammars. The pushdown automaton is a finite state machine with the added power to use a stack memory. Neural networks should be able to perform the same type of computation and thus solve such learning problems as grammatical inference [Fu 1982] .

Simple grammatical inference is defined as the problem of finding (learning) a grammar from a finite set of strings, often called the teaching sample. Recall that a grammar (phrase-structured) is defined as a 4-tuple ( N, V, P, S) where N and V are a nonterminal and terminal vocabularies, P is a finite set of production rules and S is the start symbol. Here grammatical inference is also defined as the learning of the machine that recognizes the teaching and testing samples. Potential applications of grammatical inference include such various areas as pattern recognition, information retrieval, programming language design, translation and compiling and graphics languages [Fu 1982].

There has been a great deal of interest in teaching neural nets to recognize grammars and simulate automata [Allen 1989, Jordan 1986, Pollack 1989, Servant-Schreiber et. al. 1989,Williams & Zipser 1988]. Some important extensions of that work are discussed here.  In particular we construct recurrent higher order neural net state machines which have no hidden layers and seem to be at least as powerful as any neural net multilayer state machine discussed so far. For example, the learning time and training sample size are significantly reduced.  In addition,  we integrate this neural net finite state machine with an external stack memory and inform the network through a common objective function that it has at its disposal the symbol at the top of the stack and the operation primitives of push and pop. By devising a common error function which integrates the stack and the neural net state machine, this hybrid structure learns to effectively use the stack to recognize context-free grammars.   In  the interesting work of [Williams & Zipser 1988]  a recurrent net  learns only the state machine part of a Turing Machine, since the associated move, read, write operations for each input string are known and are given as part of the training set. However, the model we present learns how to manipulate  the push, pop, and read primitives of an external stack memory plus  learns  the additional necessary state operations and  structure.

## HIGHER ORDER RECURRENT NETWORK

The recurrent neural network utilized can be considered as a higher order modification of the network model developed by [Williams & Zipser 1988].  Recall that in a recurrent net the activation state S of the neurons  at time (t+1) is defined as in a state machine automata:

$$S(t+1) = F \{ S(t), I(t); W \} \quad , \tag{1}$$

where F maps the state S and the input I at time t to the next state. The weight matrix W forms the mapping and is usually learned.  We use a higher order form for this mapping:

$$S_i(t+1) = g\{ \sum W_{ijk} S_j(t) I_k(t) \} \quad , \tag{2}$$

where the range of i, j is the number of state neurons and k the number of input neurons; g is defined as g(x)=1/(1+exp(-x)). In order to use the net for grammatical inference, a learning rule must be devised.    To learn the mapping F and the weight matrix W, given a sample set of P strings of the grammar, we construct the following error function E :

$$E = \Sigma E_r^2 = \Sigma ( T_r - S_o(t_p))^2 \qquad , \qquad (3)$$

where the sum is over P samples . The error function is evaluated at the end of a presented sequence of length $t_p$ and $S_o$ is the activity of the output neuron. For a recurrent net, the output neuron is a designated member of the state neurons. The target value of any pattern is 1 for a legal string and 0 for an illegal one. Using a gradient descent procedure, we minimize the error E function for only the rth pattern. The weight update rule becomes

$$\Delta W_{ijk} = -\eta \, \nabla_W E = \eta \, E_r \, \{ \, \partial S_o(t_p) / \partial W_{ijk} \, \} \quad , \qquad (4)$$

where $\eta$ is the learning rate. Using eq. (2), $\partial S_o(t_p) / \partial W_{ijk}$ is easily calculated using the recursion relationship and the choice of an initial value for $\partial S_i(t = 0)/\partial W_{ijk}$,

$$\partial S_l(t+1)/\partial W_{ijk} = h_l \, (S_l(t+1)) \, \{ \, \delta_{li} \, S_j(t) \, I_k(t) + \Sigma \, W_{lmn} \, I_n(t) \, \partial S_m(t)/\partial W_{ijk} \, \} \quad (5)$$

where h(x) = dg/dx. Note that this requires $\partial S_i(t) / \partial W_{ijk}$ be updated as each element of each string is presented and to have a known  initial value. Given an adequate network topology, the above neural net state machine should be capable of learning any regular grammar of arbitrary string length or a more complex grammar of finite length.

## FINITE STATE MACHINE SIMULATION

In order to see how such a net performs, we trained the net on a regular grammar, the dual parity grammar.  An arbitrary length string of 0's and 1's has dual parity if the string contains an even number of 0's and an even number of 1's. The network architecture was 3 input neurons and either 3, 4, or 5 state neurons with fully connected second order interconnection weights.  The string vocabulary 0,1,e (end symbol) used a unary representation.  The initial training set consisted of 30 positive and negative strings of increasing sting length up to length 4.  After including in the training all strings up to length 10  which resulted in misclassification(about 30 strings), the neural net state machine perfectly recognized on all strings up to length 20.  Total training time was usually 500 epochs or less.

By looking closely at the dynamics of learning, it was discovered that for different inputs  the states of the network  tended to cluster around three values plus the initial state. These four states can be considered as  possible states of an actual finite state machine and the movement between these states as a function of input can be interpreted as the state transitions of a state machine. Constructing a state machine yields a perfect four state machine which will recognize any dual parity grammar. Using minimization procedures [Fu 1982], the extraneous state transitions can be reduced to the minimal 4-

state machine. The extracted state machine is shown in Fig. 1. However, for more complicated grammars and different initial conditions, it might be difficult to extract the finite state machine. When different initial weights were chosen, different extraneous transition diagrams with more states resulted. What is interesting is that the neural net finite state machine learned this simple grammar perfectly. A first order net can also learn this problem; the higher order net learns it much faster. It is easy to prove that there are finite sate machines that cannot be represented by first order, single layer recurrent nets [Minsky 1967]. For further discussion of higher order state machines, see [Liu, et. al. 1990].

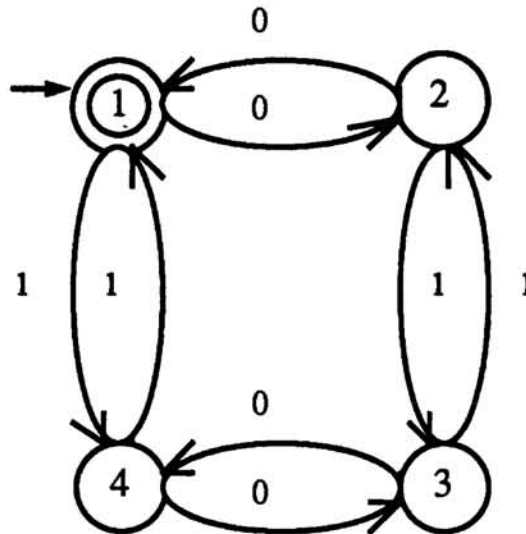

**FIGURE 1:** A learned four state machine; state 1 is both the start and the final state.

## NEURAL NET PUSHDOWN AUTOMATA

In order to easily learn more complex deterministic grammars, the neural net must somehow develop and/or learn to use some type of memory, the simplest being a stack memory. Two approaches easily come to mind. Teach the additional weight structure in a multilayer neural network to serve as memory [Pollack 1989] or teach the neural net to use an external memory source. The latter is appealing because it is well known from formal language theory that a finite stack machine requires significantly fewer resources than a finite state machine for bounded problems such as recognizing a finite length context-free grammar. To teach a neural net to use a stack memory poses at least three problems: 1) how to construct the stack memory, 2) how to couple the stack memory to the neural net state machine, and 3) how to formulate the objective function such that its optimization will yield effective learning rules.

Most straight-forward is formulating the objective function so that the stack is coupled to the neural net state machine. The most stringent condition for a pushdown automata to accept a context-free grammar is that the pushdown automata be in a final state and the stack be empty. Thus, the error function of eq. (3) above is modified to include both final state and stack length terms:

$$E = \Sigma E_r^2 = \Sigma ( T_r - S_o(t_p) + L(t_p))^2, \qquad (6)$$

where $L(t_p)$ is the final stack length at time $t_p$, i.e. the time at which the last symbol of the string is presented. Therefore, for legal strings $E = 0$, if the pushdown automata is in a final state and the stack is empty.

Now consider how the stack can be connected to the neural net state machine. Recall that for a pushdown automata [Fu 1982], the state transition mapping of eq. (1) includes an additional argument, the symbol R(t) read from the top of the stack and an additional stack action mapping. An obvious approach to connecting the stack to the neural net is to let the activity level of certain neurons represent the symbol at the top of the stack and others represent the action on the stack. The pushdown automata has an additional stack action of reading or writing to the top of the stack based on the current state, input, and top stack symbol. One interpretation of these mappings would be extensions of eq. (2):

$$S_i(t+1) = g\{ \Sigma W^s_{ijk} \; S_j(t) \; V_k(t) \} \qquad (7)$$

$$A_i(t+1) = f\{ \Sigma W^a_{ijk} \; S_j(t) \; V_k(t) \} \qquad (8)$$

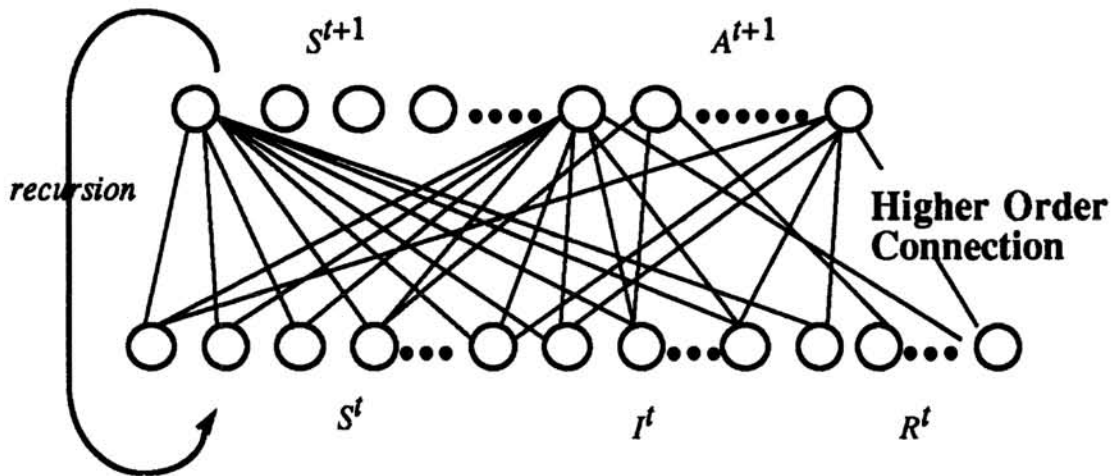

FIGURE 2:. Single layer higher order recursive neural network that is connected to a stack memory. *A* represents action neurons connected to the stack; *R* represents memory buffer neurons which read the top of the stack. The activation proceeds upward from states, input, and stack top at time t to states and action at time t+1. The recursion replaces the states in the bottom layer with the states in the top layer.

where $A_j(t)$ are output neurons controlling the action of the stack; $V_k(t)$ is either the input neuron value $I_k(t)$ or the connected stack memory neuron value $R_k(t)$, dependent on the index k; and $f=2g-1$. The current values $S_j(t)$, $I_k(t)$, and $R_k(t)$ are all fully connected through 2nd order weights with no hidden neurons. The mappings of eqs. (7) and (8) define the recursive network and can be implemented concurrently and in parallel. Let A(t=0) & R(t=0)= 0. The neuron state values range continuously from 0 to 1 while the neuron action values range from -1 to 1. The neural network part of the architecture

is depicted in Fig. 2. The number of read neurons is equal to the coding representation of the stack. For most applications, one action neuron suffices.

In order to use the gradient descent learning rule described in eq. (4), the stack length must have continuous values. (Other types of learning algorithms may not require a continuous stack.) We now explain how a continuous stack is used and connected to the action and read neurons. Interpret the stack actions as follows: push ($A>0$), pop ($A<0$), no action ($A=0$). For simplicity, only the current input symbol is pushed ; then the number of input and stack memory neurons are equal. (If the input symbol is $a$, then only A$a$ of that value is pushed into the stack) T he stack consists of a summation of analog symbols. By definition, all symbols up in unit depth one are in the read neuron R at time t.. If $A<0$ (pop), a depth of $|A|$ of all symbols in that depth is removed from the stack. In the next time step what remains in R is  a unit length from the current stack top. An attempt to pop an empty stack occurs if  not enough remains in the stack to pop depth $|A|$. Further description of this operation with examples can be found in [Sun, et. al.1990]. Since the action operation A removes or adds to the stack, the stack length at time t+1 is $L(t+1) = L(t) + A(t)$, where $L(t=0) = 0$.

With the recursion relations, stack construction, and error function defined, the learning algorithms may be derived from eqs. (4) & (6)

$$\Delta W_{ijk} = \eta \ E_r \ \{ \ \partial S_l(t_p)/\partial W_{ijk} - \partial L(t_p)/\partial W_{ij} . \tag{9}$$

The derivative terms may be derived from the recurrent relations eqs. (7) & (8) and the stack length equation. They are

$$\partial S_l(t+1)/\partial W_{ijk} = h_l \ S_l(t+1) \ \{ \ \delta_{il} \ S_j(t) \ V_k(t) + \Sigma \ W_{lmn} \ V_n(t) \ \partial S_m(t)/\partial W_{ijk} +$$
$$\Sigma \ W_{lmn} \ S_m(t) \ \partial R_n(t)/\partial W_{ijk} \ \} \tag{10}$$

and

$$\partial L(t+1)/\partial W_{ijk} = \ \partial L(t)/\partial W_{ijk} + \ \partial A(t)/\partial W_{ijk} . \tag{11}$$

Since the change $\partial R_k(t)/\partial W_{ijk}$ must contain information about past changes in action A, we have

$$\partial R_k(t)/\partial W_{ijk} = \Sigma \ \partial R_k(t)/\partial A(t) \ \partial A(t)/\partial W_{ijk} \cong \Delta_R \ \partial A(t)/\partial W_{ijk} \tag{12}$$

where  $\Delta_R = 0,1,$ or -1 and depends on the top and bottom symbols read in R(t). This approximation assumes that the read changes are only effected by actions which occurred in the recent past. The change in action with respect to the weights is defined by a recursion derived from eq. (8) and has the same form as eq. (10). For the case of popping an empty stack, the weight change increases the stack length for a legal string; otherwise nothing happens. It appears that all these derivatives are necessary to adequately integrate the neural net to the continuous stack memory.

## PUSHDOWN AUTOMATA SIMULATIONS

To test this theoretical development, we trained the neural net pushdown automaton on

two context-free grammars, $1^n0^n$ and the parenthesis grammar (balanced strings of parentheses). For the parenthesis grammar, the net architecture consisted of a 2nd order fully interconnected single layer net with 3 state neurons, 3 input neurons, and 2 action neurons (one for push & one for pop). In 20 epochs with fifty positive and negative training samples of increasing length up to length eight, the network learned how to be a perfect pushdown automaton. We concluded this after testing on all strings up to length 20 and through a similar analysis of emergent state-stack values. Using a similar clustering analysis and heuristic reduction approach, the minimal pushdown automaton emerges. It should be noted that for this pushdown automaton, the state machine does very little and is easily learned Fig. 3 shows the pushdown automaton that emerged; the 3-tuple represents (input symbol, stack symbol, action of push or pop). The $1^n0^n$ was also successfully trained with a small training set and a few hundred epochs of learning. This should be compared to the more computationally intense learning of layered networks [Allen 1989]. A minimal pushdown automaton was also derived. For further details of the learning and emergent pushdown automata, see [Sun, et.al. 1990].

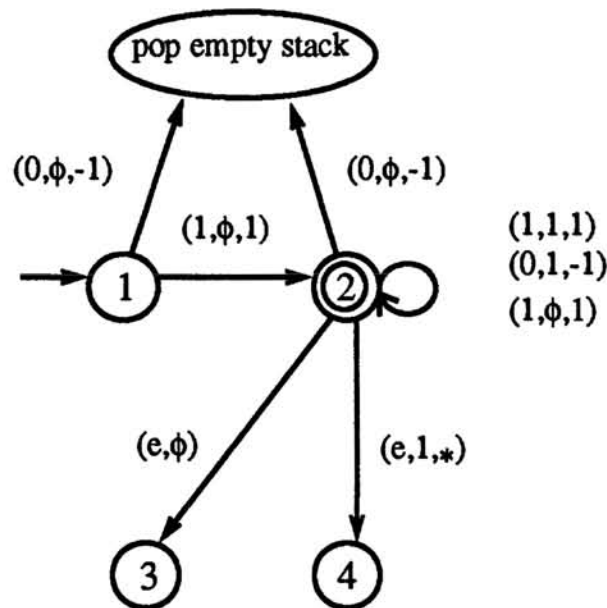

**FIGURE 3:** Learned neural network pushdown automaton for parenthesis balance checker where the numerical results for states (1), (2), (3), and (4) are (1,0,0), (.9,.2,.2), (.89,.17,.48) and (.79,.25,.70). State (1) is the start state. State (3) is a legal end state. Before feeding the end symbol, a legal string must end at state (2) with empty stack.

## CONCLUSIONS

This work presents a different approach to incorporating and using memory in a neural network. A recurrent higher order net learned to effectively employ an external stack

memory to learn simple context-free grammars. However, to do so required the creation of a continuous stack structure. Since it was possible to reduce the neural network to the ideal pushdown automaton, the neural network can be said to have "perfectly" learned these simple grammars. Though the simulations appear very promising, many questions remain. Besides extending the simulations to more complex grammars, there are questions of how well such architectures will scale for "real" problems. What became evident was the power of the higher order network; again demonstrating its speed of learning and sparseness of training sets. Will the same be true for more complex problems is a question for further work.

# REFERENCES

R.A. Allen, Adaptive Training for Connectionist State Machines, *ACM Computer Conference*, Louisville, p.428, (1989).

D. Angluin & C.H. Smith, Inductive Inference: Theory and Methods, *ACM Computing Surveys*, Vol. 15, No. 3, p. 237, (1983).

K.S. Fu, *Syntactic Pattern Recognition and Applications*, Prentice-Hall, Englewood Cliffs, N.J. (1982).

J.E. Hopcroft & J.D. Ullman, *Introduction to Automata Theory, Languages, and Computation*, Addison Wesley, Reading, Ma. (1979).

M.I. Jordan, Attractor Dynamics and Parallelism in a Connectionist Sequential Machine, *Proceedings of the Eigtht Conference of the Cognitive Science Society*, Amherst, Ma, p. 531 (1986).

Y.D. Liu, G.Z. Sun, H.H. Chen, Y.C. Lee, C.L. Giles, Grammatical Inference and Neural Network State Machines, *Proceedings of the International Joint Conference on Neural Networks*, M. Caudill (ed), Lawerence Erlbaum, Hillsdale, N.J., vol 1. p.285 (1990).

M.L. Minsky, *Computation: Finite and Infinite Machines*, Prentice-Hall, Englewood, N.J., p. 55 (1967).

F.J. Pineda, Generalization of Backpropagation to Recurrent Neural Networks, *Phys. Rev. Lett.*, vol 18, p. 2229 (1987).

J.B. Pollack, Implications of Recursive Distributed Representations, *Advances in Neural Information Systems 1*, D.S. Touretzky (ed), Morgan Kaufmann, San Mateo, Ca, p. 527 (1989).

D. Servan-Schreiber, A. Cleeremans & J.L. McClelland, Encoding Sequential Structure in Simple Recurrent Networks, *Advances in Neural Information Systems 1*, D.S. Touretzky (ed), Morgan Kaufmann, San Mateo, Ca, p. 643 (1989).

G.Z. Sun, H.H. Chen, C.L. Giles, Y.C. Lee, D. Chen, Connectionist Pushdown Automata that Learn Context-free Grammars, *Proceedings of the International Joint Conference on Neural Networks*, M. Caudill (ed), Lawerence Erlbaum, Hillsdale, N.J., vol 1. p.577 (1990).

R. J. Williams & D. Zipser, A Learning Algorithm for Continually Running Fully Recurrent Neural Networks, Institute for Cognitive Science Report 8805, U. of CA, San Diego, La Jolla, Ca 92093, (1988).
